# Spatiotemporal Coupling and Scaling of Natural Images and Human Visual Sensitivities

**Dawei W. Dong**

California Institute of Technology
Mail Code 139-74
Pasadena, CA 91125

dawei@hope.caltech.edu

## Abstract

We study the spatiotemporal correlation in natural time-varying images and explore the hypothesis that the visual system is concerned with the optimal coding of visual representation through spatiotemporal decorrelation of the input signal. Based on the measured spatiotemporal power spectrum, the transform needed to decorrelate input signal is derived analytically and then compared with the actual processing observed in psychophysical experiments.

## 1   Introduction

The visual system is concerned with the perception of objects in a dynamic world. A significant fact about natural time-varying images is that they do not change randomly over space-time; instead image intensities at different times and/or spatial positions are highly correlated. We measured the spatiotemporal correlation function – equivalently the power spectrum – of natural images and we find that it is non-separable, i.e., coupled in space and time, and exhibits a very interesting scaling behaviour. When expressed as a function of an appropriately scaled frequency variable, the spatiotemporal power spectrum is given by a simple power-law. We point out that the same kind of spatiotemporal coupling and scaling exists in human visual sensitivity measured in psychophysical experiments. This poses the intriguing question of whether there is a quantitative relationship between the power spectrum of natural images and visual sensitivity. We answer this question by showing that the latter can be predicted from measurements of the power spectrum.

## 2    Spatiotemporal Coupling and Scaling

Interest in properties of time-varying images dates back to the early days of development of the television [1]. But systematic studies have not been possible previously primarily due to technical obstacles, and our knowledge of the regularities of time-varying images has so far been very limited.

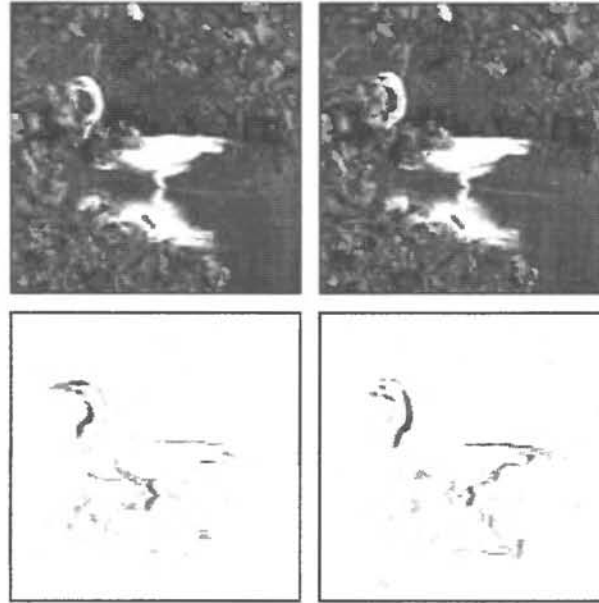

Figure 1: Natural time-varying images are highly correlated in space and time. Shown on the top are two frames of a motion scene separated by thirty three milliseconds. These two frames are highly repetitive, in fact the light intensities of most corresponding pixels are similar. Shown on the bottom are light increase (on the left) and light decrease (on the right) between the above two snapshots indicated by greyscale of pixels (white means no change). One can immediately see that only a small portion of the image changes significantly over this time scale. Our methods have been described previously [3]. To summerize, more than one thousand segments of videos on 8mm video tape (NTSC format RGB) are digitized to 8 bits greyscale using a Silicon Graphics Video board with default factory settings. Two types of segments are analyzed. The first are segments from movies on video tapes (*e.g.* "Raiders of the Lost Ark", "Uncommon Valor"). The second type of segments that we analyzed are videos made by the authors. The scene of the moving egret shown here is taken at Central Park in New York City.

We have systematically measured the two point correlation matrix or covariance matrix of $10^o \times 10^o \times 2s$ (horizontal$\times$vertical$\times$temporal digitized to $64 \times 64 \times 64$) segments of natural time-varying images by averaging over 1049 movie segments. An example of two consecutive frames from a typical segment is given in Figure 1. The Fourier transform of the correlation matrix, or the power spectrum, turns out to be a non-separable function of spatial and temporal frequencies and exhibits an interesting scaling behaviour. From our measurements (see Figure 2) we find

$$R(f, w) = R(f_w)$$

where $f_w$ is a scaled frequency which is simply the spatial frequency $f$ scaled by $G(w/f)$, a function of the ratio of temporal and spatial frequencies, i.e., $f_w = G(w/f)f$. This behaviour is revealed most clearly by plotting the power spectrum as a function of $f$ for fixed $w/f$ ratio: the curves for different $w/f$ ratios are just a horizontal shift from each other.

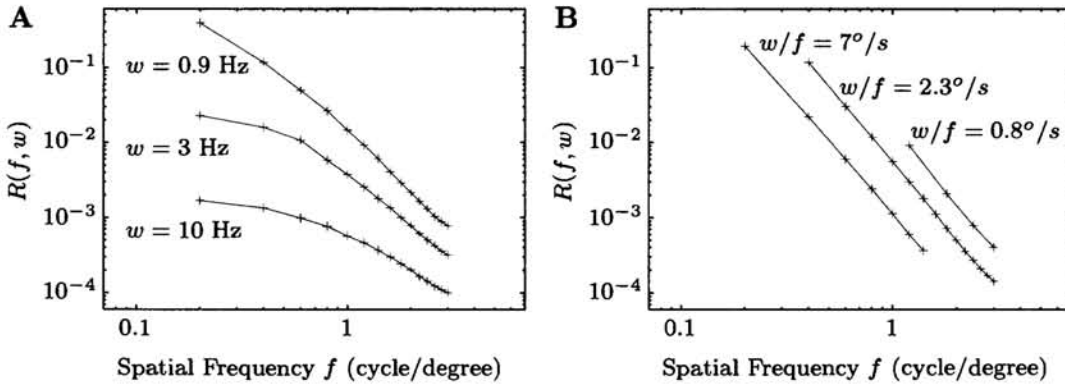

Figure 2: Spatiotemporal power spectra of natural time-varying images. (**A**) plotted as a function of spatial frequency for three temporal frequencies (0.9, 3, 10) Hz; (**B**) plotted for three velocities — ratios of temporal and spatial frequencies — (0.8, 2.3, 7) degree/second. There are some important conclusions that can be drawn from this measurement. First, it is obvious that the power spectrum cannot be separated into pure spatial and pure temporal parts; space and time are coupled in a non-trivial way. The power spectrum at low temporal frequency decreases more rapidly with increasing spatial frequency. Second, underlying this data is an interesting scaling behaviour which can be easily seen from the curves for constant $w/f$ ratios: each curve is simply shifted horizontally from each other in the log-log plot. Thus curves for constant $w/f$ ratio overlap with each other when shifted by an amount of $G(w/f)$, i.e., when plotted against a scaled frequency $f_w = G(w/f)f$. The similar spatio-temporal coupling and scaling for hunam visual sensitivity is shown in Figure 3.

Interestingly, the human visual system seems to be designed to take advantage of such regularity in natural images. The spatiotemporal contrast sensitivity of human $K(f, w)$, i.e., the visual responses to a sinewave grating of spatial frequency $f$ modulated at temporal frequency $w$, exhibits the same kind of spatiotemporal coupling and scaling (see Figure 3),

$$K(f, w) = K(f_w).$$

Again, when the contrast sensitivity curves are plotted as a function of $f$ for fixed $w/f$ ratios, the curves have the same shape and are only shifted from each other [2].

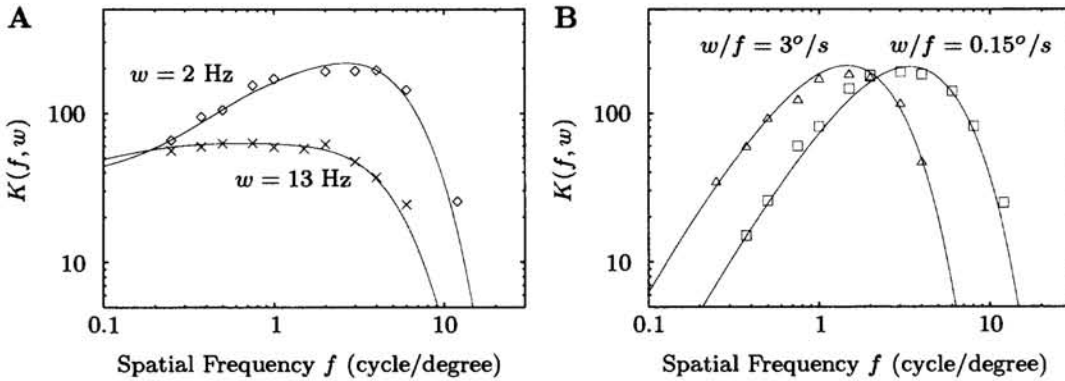

Figure 3: Spatiotemporal contrast sensitivities of human vision. (**A**) plotted as a function of spatial frequency for two temporal frequencies (2, 13) Hz; (**B**) plotted for two $w/f$ ratios (0.15, 3) degree/second. The solid lines in both A and B are the empirical fits. The experimental data points and empirical fitting curves are from reference [2]. First, it can be seen that the human visual sensitivity curve is band-pass filter at low temporal frequency and approaches low-pass filter for higher temporal frequency. The space and time are coupled. Second, it is clear that the curves for different $w/f$ ratios have the same shape and are only shifted horizontally from each other in the log-log plot. Again, curves for constant $w/f$ ratio overlap with each other when shifted by an amount of $G(w/f)$, i.e., when plotted against a scaled frequency $f_w = G(w/f)f$. The similar behaviour of spatiotemporal coupling and scaling for the power spectra of natural images is shown in Figure 2.

## 3   Relative Motion of Visual Scene

Why does the human visual sensitivity have the same spatiotemporal coupling and scaling as natural images?

The intuition underlying the spatiotemporal coupling and scaling of natural images is that when viewing a real visual scene the natural eye and/or body movements translate the entire scene across the retina and every spatial Fourier component of the scene moves at the same velocity. Thus it is reasonable to assume that for constant velocity, i.e., $w/f$ ratio, the power spectrum show the same universal behaviour. This assumption is tested quantitatively in the following.

Our measurements reveal that the spatiotemporal power spectrum has a simple form

$$R(f_w) \sim f_w^{-3}$$

which is shown in Figure 6A. This behaviour can be accounted for if the dominant component in the temporal signal comes from motion of objects with static power spectra of $R_s(f) \sim f^{-2}$. The static power spectra for the same collection of images is measured by treating frames as snapshots (Figure 4A); the measurement confirmed the above assumption and is in agreement with earlier works on the statistical properties of static natural images [5, 6, 7].

It is easy to derive that for a rotationally symmetric static spectrum $R_s(f) = K/f^2$ ($K$ is a constant), the spatiotemporal power spectrum of moving images is

$$R(f, w) = \frac{K}{f^3} P(\frac{w}{f}),  \qquad (1)$$

where $P(\frac{w}{f})$ is the function of velocity distribution, which is shown as the solid curve in Figure 4B (measured independently from the optical flows between frames).

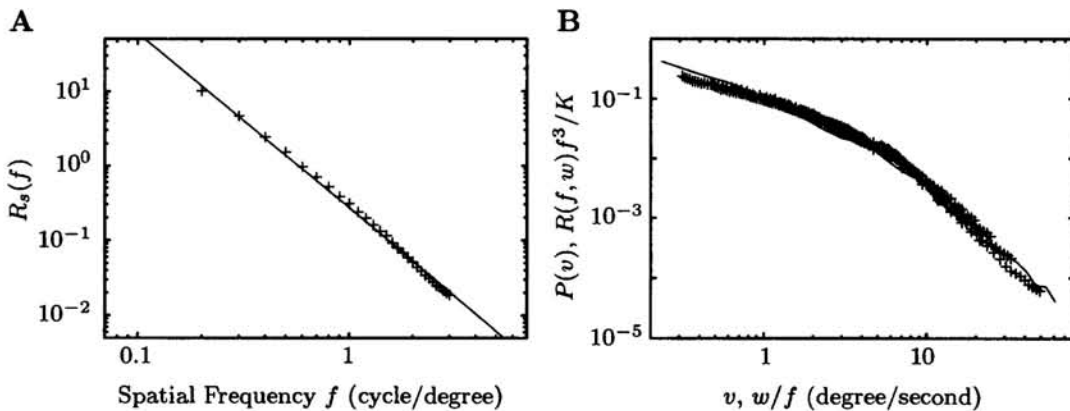

Figure 4: Spatial power spectrum and velocity distribution. **(A)** the measured spatial power spectrum of snap shot images, which shows that $R_s(f) \sim K/f^2$ is a good approximation to the spectrum; **(B)** the measured velocity distribution $P(v)$ (solid curve), in which the data of Figure 2 for the power spectrum were replotted as a function of $w/f$ after multiplication by $f^3$ — all the data points fall on the $P(v)$ curve.

In summary, the measured spatiotemporal power spectrum is dominated by images of spatial power spectrum $\sim 1/f^2$ moving with a velocity distribution $P(v) \sim 1/(v + v_0)^2$ (similar velocity distribution has been proposed earlier [8, 3] . Thus $R(f, w) = K/f^3(w/f + v_0)^2$ and $G(w/f) \sim (w/f + v_0)^{2/3}$.

Based on the assumption that the visual system is optimized to transmit information from natural scenes, we have derived and pointed out in references [3, 4] that the spatiotemporal contrast sensitivity $K$ is a function of the power spectrum $R$, and thus the spatiotemporal coupling and scaling of $R$ of natural images translates directly to the spatiotemporal coupling and scaling of $K$ of visual sensitivity i.e., $R$ is a function of $f_w$ only, so is $K$.

## 4  Spatiotemporal Decorrelation

The theory of spatiotemporal decorrelation is based on ideas of optimal coding from information theory: decorrelation of inputs to make statistically independent representations when signal is strong and smoothing where noise is significant. The end result is that by chosing the correct degree of decorrelation the signal is compressed by elimination of what is irrelevant without significant loss of information.

The following relationship can be derived for the visual sensitivity $K$ and the power spectrum $R$ in the presence of noise power $N$:

$$K = R^{-1/2}(1 + N/R)^{-3/2}$$

The figure below illustrates the predicted filter for the case of white noise (constant $N$).

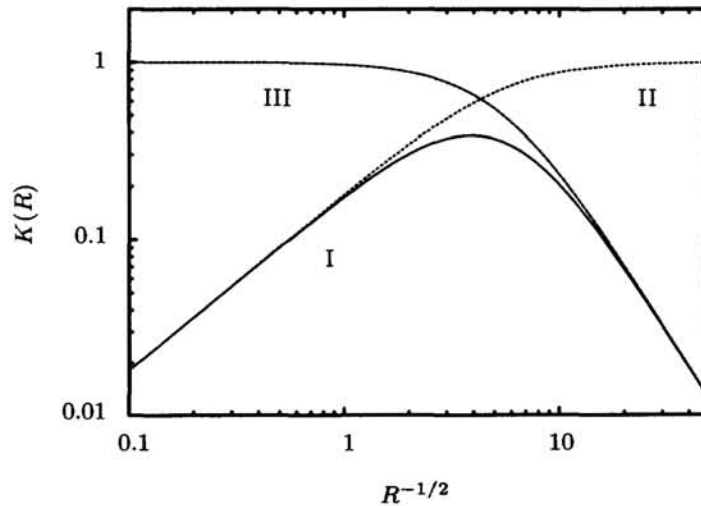

Figure 5: Predicted optimal filter (curve I): in the low noise regime, it is given by whitening filter $R^{-1/2}$ (curve II), which achieves spatiotemporal decorrelation; while at high noise regime it asymptotes the low-pass filter (curve III) which suppresses noise.

As shown in Figure 6, the relation between the contrast sensitivity and the power spectrum predicts

$$K(f_w) \sim \left( \frac{f_w}{1 + N f_w^3} \right)^{\frac{3}{2}}$$

in which $N$ is the power of the white noise. This prediction is compared with psychophysical data in Figure 6B where we have used the scaling function $G(w/f) = (w/f + v_0)^{2/3}$ which has the same asymptotic behaviour as we have shown for the natural time-varying images [3]. We find that for $v_0 = 1$ degree/second, the human

contrast sensitivity curves for $w/f$ from 0.1 to 4 degree/second, measured in reference [2], overlap very well with the theoretical prediction from the power spectrum of our measurements.

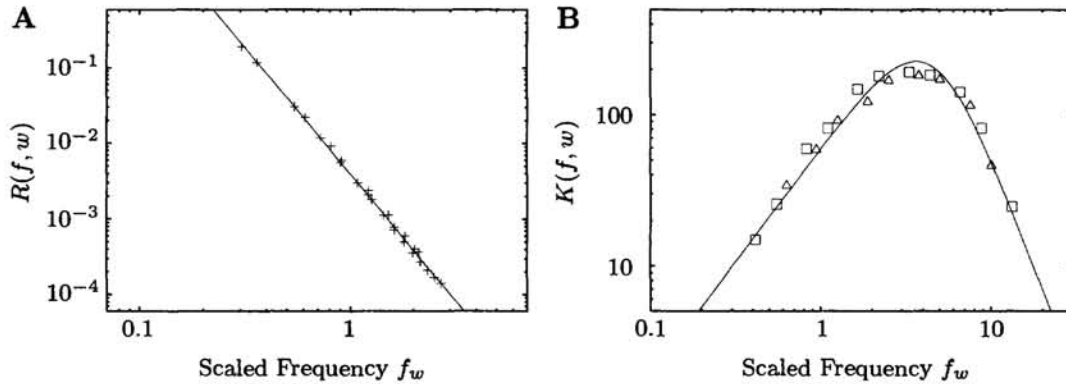

Figure 6: Relation between the power spectrum of natural images and the human visual sensitivities. (**A**) the measured spatiotemporal power spectrum (Figure 2B) replotted as a function of the scaled frequency can be fit very well by $R \sim f_w^{-3}$ (solid line); (**B**) the spatiotemporal contrast sensitivities of human vision (Figure 3B) replotted as a function of the scaled frequency can be fit very well by our theoretical prediction (solid line). Our theory on the relation between the visual sensitivity $K$ and the power spectrum of natural time-varying images $R$ in the presence of noise power $N$ has been described in detail in reference [4]. To summarize, the visual sensitivity in Fourier space is simply $K = R^{-1/2}(1 + N/R)^{-3/2}$. In a linear system, this is proportional to the visual response to a sinewave of spatial frequency $f$ modulated at temporal frequency $w$, i.e., the contrast sensitivity curves shown in Figure 3. In the case of white noise, i.e., $N$ is independent of $f$ and $w$, $K$ depends on $f$ and $w$ through the power spectrum $R$. Since $R$ is a function of the scaled frequency $f_w$ only, so is $K$. From our measurement $R \sim f_w^3$, thus $K \sim f_w^{3/2}(1 + Nf_w^3)^{-3/2}$. This curve is plotted in the figure as the solid line with $N = 0.01$. The agreement is very good.

## 5   Conclusions and Discussions

A simple relationship is revealed between the statistical structure of natural time-varying images and the spatiotemporal sensitivity of human vision. The existence of this relationship supports the hypothesis that visual processing is optimized to compress as much information as possible about the outside world into the limited dynamic range of the visual channels.

We should point out that this scaling behaviour is expected to break down for very high temporal and spatial frequency where the effect of the temporal and spatial modulation function of the eye [9, 10] cannot be ignored.

Finally while our predictions show that, in general, the human visual sensitivity is strongly space-time coupled, we do predict a regime where decoupling is a good approximation. This is based on the fact that in the regime of relatively high temporal frequency and relatively low spatial frequency we find that the power spectrum of natural images is separable into spatial and temporal parts [3]. In a previous work we have used this decoupling to model response properties of cat LGN cells where we have shown that these can be accounted for by the theoretical prediction based on the power spectrum in that regime [4].

## Acknowledgements

The author gratefully acknowledges the discussions with Dr. Joseph Atick.

# References

[1] Kretzmer ER, 1952. Statistics of television signals. The bell system technical journal. 751-763.

[2] Kelly DH, 1979 Motion and vision. II. Stabilized spatio-temporal threshold surface. J. Opt. Soc. Am. 69, 1340-1349.

[3] Dong DW, Atick JJ, 1995 Statistics of natural time-varying images. Network: Computation in Neural Systems, 6, 345-358.

[4] Dong DW, Atick JJ, 1995 Temporal decorrelation: a theory of lagged and nonlagged responses in the lateral geniculate nucleus. Network: Computation in Neural Systems, 6, 159-178.

[5] Burton GJ, Moorhead IR, 1987. Color and spatial structure in natural scenes. Applied Optics. 26(1): 157-170.

[6] Field DJ, 1987. Relations between the statistics of natural images and the response properties of cortical cells.. J. Opt. Soc. Am. A 4: 2379-2394.

[7] Ruderman DL, Bialek W, 1994. Statistics of natural images: scaling in the woods. Phy. Rev. Let. 73(6): 814-817.

[8] Van Hateren JH, 1993. Spatiotemporal Contrast sensitivity of early vision. Vision Res. 33(2): 257-267.

[9] Campbell FW, Gubisch RW, 1966. Optical quality of the human eye. J. Physiol. 186: 558-578.

[10] Schnapf JL, Baylor DA, 1987. How photoreceptor cells respond to light. Scientific American 256(4): 40-47.
